# Occlusive Components Analysis

**Jörg Lücke**
Frankfurt Institute for Advanced Studies
Goethe-University Frankfurt, Germany
`luecke@fias.uni-frankfurt.de`

**Richard Turner**
Gatsby Computational Neuroscience Unit, UCL
17 Queen Square, London WC1N 3AR, UK
`turner@gatsby.ucl.ac.uk`

**Maneesh Sahani**
Gatsby Computational Neuroscience Unit, UCL
17 Queen Square, London WC1N 3AR, UK
`maneesh@gatsby.ucl.ac.uk`

**Marc Henniges**
Frankfurt Institute for Advanced Studies
Goethe-University Frankfurt, Germany
`henniges@fias.uni-frankfurt.de`

## Abstract

We study unsupervised learning in a probabilistic generative model for occlusion. The model uses two types of latent variables: one indicates which objects are present in the image, and the other how they are ordered in depth. This depth order then determines how the positions and appearances of the objects present, specified in the model parameters, combine to form the image. We show that the object parameters can be learnt from an unlabelled set of images in which objects occlude one another. Exact maximum-likelihood learning is intractable. However, we show that tractable approximations to Expectation Maximization (EM) can be found if the training images each contain only a small number of objects on average. In numerical experiments it is shown that these approximations recover the correct set of object parameters. Experiments on a novel version of the bars test using colored bars, and experiments on more realistic data, show that the algorithm performs well in extracting the generating causes. Experiments based on the standard bars benchmark test for object learning show that the algorithm performs well in comparison to other recent component extraction approaches. The model and the learning algorithm thus connect research on occlusion with the research field of multiple-causes component extraction methods.

## 1 Introduction

A long-standing goal of unsupervised learning on images is to be able to learn the shape and form of objects from unlabelled scenes. Individual images usually contain only a small subset of all possible objects. This observation has motivated the construction of algorithms—such as sparse coding (SC; [1]) or non-negative matrix factorization (NMF; [2]) and its sparse variants—based on learning in latent-variable models, where each possible object, or part of an object, is associated with a variable controlling its presence or absence in a given image. Any individual "hidden cause" is rarely active, corresponding to the small number of objects present in any one image. Despite this plausible motivation, these algorithms make severe approximations. Perhaps the most crucial is that in the underlying latent variable models, objects or parts thereof, combine *linearly* to form the image. In real images the combination of individual objects depends on their relative distance from the camera or eye. If two objects occupy the same region in planar space, the nearer one occludes the other, i.e., the hidden causes non-linearly compete to determine the pixel values in the region of overlap.

In this paper we extend multiple-causes models such as SC or NMF to handle occlusion. The idea of using many hidden "cause" variables to control the presence or absence of objects is retained, but these variables are augmented by another set of latent variables which determine the relative

depth of the objects, much as in the z-buffer employed by computer graphics. In turn, this enables the simplistic linear combination rule to be replaced by one in which nearby objects occlude those that are more distant. One of the consequences of moving to a richer, more complex model is that inference and learning become correspondingly harder. One of the main contributions of this paper is to show how to overcome these difficulties.

The problem of occlusion has been addressed in different contexts [3, 4, 5, 6]. Prominent probabilistic approaches [3, 4] assign pixels in multiple images taken from the same scene to a fixed number of image layers. The approach is most frequently applied to automatically remove foreground and background objects. Those models are in many aspects more general than the approach discussed here. However, they model, in contrast to our approach, data in which objects maintain a fixed position in depth relative to the other objects.

## 2 A Generative Model for Occlusion

The occlusion model contains three important elements. The first is a set of variables which controls the presence or absence of objects in a particular image (this part will be analogous, e.g., to NMF). The second is a variable which controls the relative depths of the objects that are present. The third is the combination rule which describes how closer active objects occlude more distant ones.

To model the presence or absence of an object we use $H$ binary hidden variables $s_1, \ldots, s_H$. We assume that the presence of one object is independent of the presence of the others and assume, for simplicity, equal probabilities $\pi$ for objects to be present:

$$p(\vec{s} \,|\, \pi) = \prod_{h=1}^{H} \text{Bernoulli}(s_h; \pi) = \prod_{h=1}^{H} \pi^{s_h} (1 - \pi)^{1-s_h}. \tag{1}$$

Objects in a real image can be ordered by their depth and it is this ordering which determines which of two overlapping objects occludes the other. The depth-ordering is captured in the model by randomly and uniformly choosing a member $\hat{\sigma}$ of the set $\mathcal{G}(|\vec{s}|)$ which contains all permutation functions $\hat{\sigma} : \{1, \ldots, |\vec{s}|\} \to \{1, \ldots, |\vec{s}|\}$, with $|\vec{s}| = \sum_h s_h$. More formally, the probability of $\hat{\sigma}$ given $\vec{s}$ is defined by:

$$p(\hat{\sigma} \,|\, \vec{s}) = \frac{1}{|\vec{s}|!} \quad \text{with} \quad \hat{\sigma} \in \mathcal{G}(|\vec{s}|). \tag{2}$$

Note that we could have defined the order in depth independently of $\vec{s}$, by choosing from $\mathcal{G}(H)$ with $p(\hat{\sigma}) = \frac{1}{H!}$. But then, because the depth of absent objects ($s_h = 0$) is irrelevant, no more than $|\vec{s}|!$ distinct choices of $\hat{\sigma}$ would have resulted in different images.

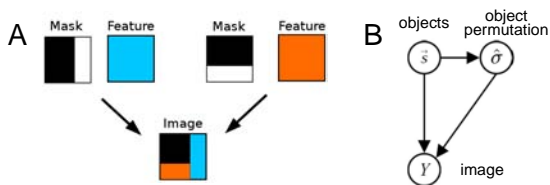

Figure 1: **A** Illustration of how two object masks and features combine to generate an image (generation without noise). **B** Graphical model of the generation process with hidden permutation variable $\hat{\sigma}$.

The final stage of the generative model describes how to produce the image given a selection of active causes and an ordering in relative depth of these causes. One approach would be to choose the closest object and to set the image equal to the feature vector associated with this object. However, this would mean that every image generated from the model would comprise just one object; the closest. What is missing from this description is a notion of the extent of an object and the fact that it might only contribute to a local selection of pixels in an image. For this reason, our model contains two sets of parameters. One set of parameters, $W \in \mathbb{R}^{H \times D}$, describes what contribution an object makes to each pixel ($D$ is the number of pixels). The vector $(W_{h1}, \ldots, W_{hD})$ is therefore described as the *mask* of object $h$. If an object is highly localized, this vector will contain many zero elements. The other set of paramenters, $T \in \mathbb{R}^{H \times C}$, represent the features of the objects. A feature vector $\vec{T}_h \in \mathbb{R}^C$ describing object $h$ might, for instance, be the object's rgb-color ($C = 3$ in that case). Fig. 1A illustrates the combination of masks and features, and Fig. 1B shows the graphical model of the generation process.

Let us formalize how an image is generated given the parameters $\Theta = (W, T)$ and given the hidden variables $S = (\vec{s}, \hat{\sigma})$. Before we consider observation noise, we define the generation of a noiseless

image $\vec{\mathcal{T}}(S,\Theta)$ to be given by:

$$\vec{\mathcal{T}}_d(S,\Theta) = W_{h_o d}\,\vec{T}_{h_o}$$
$$\text{where } h_o = \text{argmax}_h\{\tau(S,h)\,W_{hd}\}, \qquad \tau(S,h) = \begin{cases} 0 & \text{if } s_h = 0 \\ \frac{3}{2} & \text{if } s_h = 1 \text{ and } |\vec{s}| = 1 \\ \frac{\hat{\sigma}(h)-1}{|\vec{s}|-1} + 1 & \text{otherwise} \end{cases} \quad (3)$$

In (3) the order in depth is represented by the mapping $\tau$ whose specific form will facilitate later algebraic steps. To illustrate the combination rule (3) and the mapping $\tau$ consider Fig. 1A and Fig. 2. Let us assume that the mask values $W_{hd}$ are zero or one (although we will later also allow for continuous values). As depicted in Fig. 1A an object $h$ with $s_h = 1$ occupies all image pixels with $W_{hd} = 1$ and does not occupy pixels with $W_{hd} = 0$. For all pixels with $W_{hd} = 1$ the vector $\vec{T}_h$ sets the pixels' values to a specific feature, e.g., to a specific color. The function $\tau$ maps all causes $h$ with $s_h = 0$ to zero while all other causes are mapped to values within the interval $[1,2]$ (see Fig. 2). $\tau$ assigns a proximity value $\tau(S,h) > 0$ to each present object. For a given pixel $d$ the

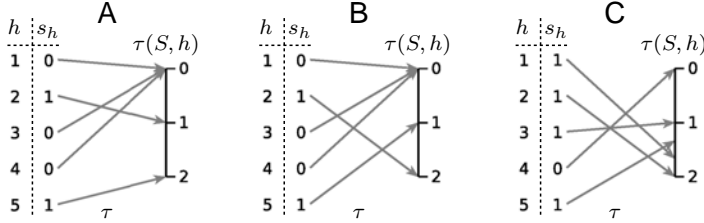

Figure 2: Visualization of the mapping $\tau$. **A** and **B** show the two possible mappings for two causes, and **C** shows one possible mapping for four causes.

combination rule (3) simply states that of all objects with $W_{hd} = 1$, the most proximal is used to set the pixel property. Given the latent variables and the noiseless image $\vec{\mathcal{T}}(S,\Theta)$, we take the observed variables $Y = (\vec{y}_1,\ldots,\vec{y}_D)$ to be drawn independently from a Gaussian distribution (which is the usual choice for component extraction systems):

$$p(Y\,|\,S,\Theta) = \prod_{d=1}^{D} p(\vec{y}_d\,|\,\vec{\mathcal{T}}_d(S,\Theta)), \qquad p(\vec{y}\,|\,\vec{t}) = \mathcal{N}(\vec{y};\vec{t},\sigma^2\mathbb{1})\,. \quad (4)$$

Equations (1) to (4) represent a generative model for occlusion.

## 3  Maximum Likelihood

One approach to learning the parameters $\Theta = (W,T)$ of this model from data $\mathcal{Y} = \{Y^{(n)}\}_{n=1,\ldots,N}$ is to use Maximum Likelihood learning, that is,

$$\Theta^* = \text{argmax}_\Theta\{\mathcal{L}(\Theta)\} \quad \text{with} \quad \mathcal{L}(\Theta) = \log\big(p(Y^{(1)},\ldots,Y^{(N)}\,|\,\Theta)\big)\,. \quad (5)$$

However, as there is usually a large number of objects that can potentially be present in the training images, and as the likelihood involves summing over all combinations of objects and associated orderings, the computation of (5) is typically intractable. Moreover, even if it were tractably computable, optimization of the likelihood is made problematic by an analytical intractability arising from the fact that the occlusion non-linearity is non-differentiable. The following section describes how to side-step the computational intractability within the standard Expectation Maximisation (EM) formalism for maximum likelihood learning, using a truncated expansion of sums for the sufficient statistics. Furthermore, as the M-Step of EM requires gradients to be computed, the section also describes how to side-step the analytical intractability by an approximate version of the model's non-linearity.

To find the parameters $\Theta^*$ at least approximately, we use the variational EM formalism (e.g., [7]) and introduce the free-energy function $\mathcal{F}(\Theta,q)$ which is a function of $\Theta$ and an unknown distribution $q(S^{(1)},\ldots,S^{(N)})$ over the hidden variables. $\mathcal{F}(\Theta,q)$ is a lower bound of the likelihood $\mathcal{L}(\Theta)$. Approximations introduced later on can be interpreted as choosing specific functions $q$, although (for brevity) we will not make this relation explicit. In the model described above, in which each image is drawn independently and identically, $q(S^{(1)},\ldots,S^{(N)}) = \prod_n q_n(S^{(n)},\Theta')$ which is taken to be parameterized by $\Theta'$. The free-energy can thus be written as:

$$\mathcal{F}(\Theta,q) = \sum_{n=1}^{N}\left[\sum_S q_n(S,\Theta')\left[\log\big(p(Y^{(n)}\,|\,S,\Theta)\big) + \log\big(p(S\,|\,\Theta)\big)\right]\right] + H(q), \quad (6)$$

where the function $H(q) = -\sum_n \sum_S q_n(S, \Theta') \log(q_n(S, \Theta'))$ (the Shannon entropy) is independent of $\Theta$. Note that $\sum_S$ in (6) sums over all possible states of $S = (\vec{s}, \hat{\sigma})$, i.e., over all binary vectors and all associated permutations in depth. This is the source of the computational intractability. In the EM scheme $\mathcal{F}(\Theta, q)$ is maximized alternately with respect to the distribution, $q$, in the E-step (while the parameters, $\Theta$, are kept fixed) and with respect to parameters, $\Theta$, in the M-step (while $q$ is kept fixed). It can be shown that an EM iteration increases the likelihood or leaves it unchanged. In practical applications EM is found to increase the likelihood to likelihood maxima, although these can be local.

**M-Step.** The M-Step of EM, in which the free-energy, $\mathcal{F}$, is optimized with respect to the parameters, is canonically derived by taking derivatives of $\mathcal{F}$ with respect to the parameters. Unfortunately, this standard procedure is not directly applicable because of the non-linear nature of occlusion as reflected by the combination rule (3). However, it is possible to approximate the combination rule by the differentiable function,

$$\vec{\mathcal{T}}^\rho{}_d(S, \Theta) \quad := \quad \frac{\sum_{h=1}^H (\tau(S,h)\, W_{hd})^\rho\, W_{hd}\, \vec{T}_h}{\sum_{h=1}^H (\tau(S,h)\, W_{hd})^\rho}. \tag{7}$$

Note that for $\rho \to \infty$ the function $\vec{\mathcal{T}}^\rho{}_d(S, \Theta)$ is equal to the combination rule in (3). $\vec{\mathcal{T}}^\rho{}_d(S, \Theta)$ is differentiable w.r.t. the parameters $W_{hd}$ and $T_h^c$ ($c \in \{1, \dots, C\}$) and it applies for large $\rho$:

$$\begin{aligned}
\frac{\partial}{\partial W_{id}} \vec{\mathcal{T}}^\rho{}_d(S, \Theta) &\approx \mathcal{A}_{id}^\rho(S, W)\, \vec{T}_i, \\
\frac{\partial}{\partial T_i^c} \vec{\mathcal{T}}^\rho{}_d(S, \Theta) &\approx \mathcal{A}_{id}^\rho(S, W)\, W_{id}\, \vec{e}_c,
\end{aligned} \quad \text{with} \quad
\begin{aligned}
\mathcal{A}_{id}^\rho(S, W) &:= \frac{(\tau(S,i)\, W_{id})^\rho}{\sum_{h=1}^H (\tau(S,h)\, W_{hd})^\rho}, \\
\mathcal{A}_{id}(S, W) &:= \lim_{\rho \to \infty} \mathcal{A}_{id}^\rho(S, W),
\end{aligned} \tag{8}$$

where $\vec{e}_c$ is a unit vector in feature space with entry 1 at position $c$ and zero elsewhere (the approximations on the left-hand-side above become equalities for $\rho \to \infty$). We can now compute approximations to the derivatives of $\mathcal{F}(\Theta, q)$. For large values of $\rho$ the following holds:

$$\frac{\partial}{\partial W_{id}} \mathcal{F}(\Theta, q) \quad \approx \quad \sum_{n=1}^N \left[ \sum_S q_n(S, \Theta') \left( \frac{\partial}{\partial W_{id}} \vec{\mathcal{T}}^\rho{}_d(S, \Theta) \right)^T \vec{f}\left( \vec{y}^{(n)}, \vec{\mathcal{T}}^\rho{}_d(S, \Theta) \right) \right], \tag{9}$$

$$\frac{\partial}{\partial T_i^c} \mathcal{F}(\Theta, q) \quad \approx \quad \sum_{n=1}^N \left[ \sum_S q_n(S, \Theta') \sum_{d=1}^D \left( \frac{\partial}{\partial T_i^c} \vec{\mathcal{T}}^\rho{}_d(S, \Theta) \right)^T \vec{f}\left( \vec{y}^{(n)}, \vec{\mathcal{T}}^\rho{}_d(S, \Theta) \right) \right], \tag{10}$$

$$\text{where} \quad \vec{f}(\vec{y}^{(n)}, \vec{t}) := \frac{\partial}{\partial \vec{t}} \log\left( p(\vec{y}^{(n)} \,|\, \vec{t}) \right) = -\sigma^{-2}\, (\vec{y}^{(n)} - \vec{t}).$$

Setting the derivatives (9) and (10) to zero and inserting equations (8) yields the following necessary conditions for a maximum of the free energy that hold in the limit $\rho \to \infty$:

$$W_{id} = \frac{\sum_n \langle \mathcal{A}_{id}(S, W) \rangle_{q_n}\, \vec{T}_i^T\, \vec{y}_d^{(n)}}{\sum_n \langle \mathcal{A}_{id}(S, W) \rangle_{q_n}\, \vec{T}_i^T\, \vec{T}_i}, \quad \vec{T}_i = \frac{\sum_n \sum_d \langle \mathcal{A}_{id}(S, W) \rangle_{q_n}\, W_{id}\, \vec{y}_d^{(n)}}{\sum_n \sum_d \langle \mathcal{A}_{id}(S, W) \rangle_{q_n}\, (W_{id})^2}. \tag{11}$$

Note that equations (11) are not straight-forward update rules. However, we can use them in the fixed-point sense and approximate the parameters which appear on the right-hand-side of the equations using the values from the previous iteration.

Equations (11), together with the exact posterior $q_n(S, \Theta') = p(S \,|\, \vec{y}^{(n)}, \Theta')$, represent a maximum-likelihood based learning algorithm for the generative model (1) to (4). Note, however, that due to the multiplication of the weights and the mask, $W_{hd}\, \vec{T}_h$ in (3), there is degeneracy in the parameters: given $h$ the combination $\vec{\mathcal{T}}_d$ remains unchanged for the operation $\vec{T}_h \to \alpha \vec{T}_h$ and $W_{hd} \to W_{hd}/\alpha$ with $\alpha \neq 0$. To remove the degeneracy we set after each iteration:

$$W_{hd}^{\text{new}} = W_{hd} / \overline{W}_h, \quad \vec{T}_h^{\text{new}} = \overline{W}_h \vec{T}_h, \text{ where } \overline{W}_h = \sum_{d \in \mathcal{I}} W_{hd} \text{ with } \mathcal{I} = \{d \,|\, W_{id} > 0.5\}. \tag{12}$$

For reasons that will briefly be discussed later, the use of $\overline{W}_h$ instead of, e.g., $W_h^{\max} = \max_d\{W_{hd}\}$ is advantageous for some data, although for many other types of data $W_h^{\max}$ works equally well.

**E-Step.** The crucial entities that have to be computed for update equations (11) are the sufficient statistics $\langle \mathcal{A}_{id}(S,W) \rangle_{q_n}$, i.e., the expectation of the function $\mathcal{A}_{id}(S,W)$ in (8) over the distribution of hidden states $S$. In order to derive a computationally tractable learning algorithm the expectation $\langle \mathcal{A}_{id}(S,W) \rangle_{q_n}$ is re-written and approximated as follows,

$$\langle \mathcal{A}_{id}(S,W) \rangle_{q_n} = \frac{\sum\limits_{S} p(S, Y^{(n)} \,|\, \Theta') \, \mathcal{A}_{id}(S,W)}{\sum\limits_{\tilde{S}} p(\tilde{S}, Y^{(n)} \,|\, \Theta')} \approx \frac{\sum\limits_{S,(|\vec{s}| \leq \chi)} p(S, Y^{(n)} \,|\, \Theta') \, \mathcal{A}_{id}(S,W)}{\sum\limits_{\tilde{S},(|\vec{\tilde{s}}| \leq \chi)} p(\tilde{S}, Y^{(n)} \,|\, \Theta')}. \quad (13)$$

That is, in order to approximate (13), the problematic sums in the numerator and denominator have been truncated. We only sum over states $\vec{s}$ with less or equal $\chi$ non-zero entries. Approximation (13) replaces the intractable exact E-step by one whose computational cost scales only polynomially with $H$ (roughly cubically for $\chi = 3$). As for other approximate EM approaches, there is no guarantee that this approximation will always result in an increase of the data likelihood. For data points that were generated by a small number of causes on average we can, however, expect the approximation to match an exact E-step with increasing accuracy the closer we get to the optimum. For reasons highlighted earlier, such data will be typical in image modelling. A truncation approach similar to (13) has successfully been used in the context of the maximal causes generative model in [8]. Also in the case of occlusion we will later see that in numerical experiments using approximation (13) the true generating causes are indeed recovered.

## 4 Experiments

In order to evaluate the algorithm it has been applied to artificial data, where its performance can be compared to ground truth, and to more realistic visual data. In all the experiments we use image pixels as input variables $\vec{y}_d$. The entries of the observed variables $\vec{y}_d$ are set by the pixels' rgb-color vector, $\vec{y}_d \in [0,1]^3$. In all trials of all experiments the initial values of the mask parameters $W_{hd}$ and the feature parameters $T_h^c$ were independently and uniformly drawn from the interval $[0,1]$.

**Learning and annealing.** The free-energy landscape traversed by EM algorithms is often multi-modal. Therefore EM algorithms can converge to local optima. However, this problem can be alleviated using deterministic annealing as described in [9, 10]. For the model under consideration here annealing amounts to the substitutions $\pi \rightarrow \pi^\beta$, $(1-\pi) \rightarrow (1-\pi)^\beta$, and $(1/\sigma^2) \rightarrow (\beta/\sigma^2)$, with $\beta = 1/\hat{T}$ in the E-step equations. During learning, the 'temperature' parameter $\hat{T}$ is decreased from an initial value $\hat{T}^{\text{init}}$ to 1. To update the parameters $W$ and $T$ we applied the M-step equations (11). For the sufficient statistics $\langle \mathcal{A}_{id}(S,W) \rangle_{q_n}$ we used approximation (13) with $\mathcal{A}_{id}^\rho(S,W)$ in (8) instead of $\mathcal{A}_{id}(S,W)$ and with $\chi = 3$ if not stated otherwise. The parameter $\rho$ was increased during learning with $\rho = \frac{1}{1-\beta}$ (with a maximum of $\rho = 20$ to avoid numerical instabilities). In all experiments we used 100 EM iterations and decreased $\hat{T}$ linearly except for 10 initial iterations at $\hat{T} = \hat{T}^{\text{init}}$ and 20 final iterations at $\hat{T} = 1$. In addition to annealing, a small amount of independent and identically distributed Gaussian noise (standard deviation 0.01) was added to the masks and the features, $W_{hd}$ and $T_d^c$, to help escape local optima. This parameter noise was linearly decreased to zero during the last 20 iterations of each trial.

**The colored bars test.** The component extraction capabilities of the model were tested using the colored bars test. This test is a generalization of the classical bars test [11] which has become a popular benchmark task for non-linear component extraction. In the standard bars test with $H = 8$ bars the input data are 16-dimensional vectors, representing a $4 \times 4$ grid of pixels, i.e., $D = 16$. The single bars appear at the 4 vertical and 4 horizontal positions. For the colored bars test, the bars have colors $\vec{T}_h^{\text{gen}}$ which are independently and uniformly drawn from the rgb-color-cube $[0,1]^3$. Once chosen, they remain fixed for the generation of the data set. For each image a bar appears independently with a probability $\pi = \frac{2}{8}$ which results in two bars per image on average (the standard value in the literature). For the bars active in an image, a ranking in depth is randomly and uniformly chosen from the permutation group. The color of each pixel is determined by the least distant bar and is black if the pixel is occupied by no bar. $N = 500$ images were generated for learning and Fig. 3A shows a random selection of 13 examples. The learning algorithms were applied to the colored bars test with $H = 8$ hidden units and $D = 16$ input units. The observation noise was set

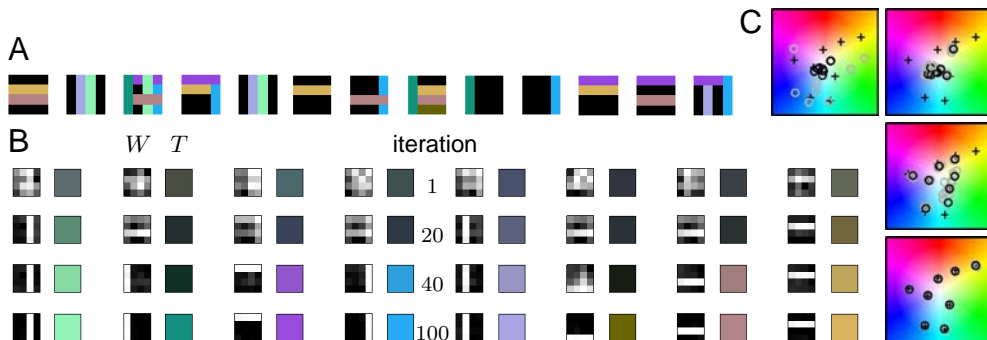

Figure 3: Application to the colored bars test. **A** Selection of 13 of the $N = 500$ data points used for learning. **B** Changes of the parameters $W$ and $T$ for the algorithm with $H = 8$ hidden units. Each row shows $W$ and $T$ for the specified EM iteration. **C** Feature vectors at the iterations in **B** displayed as points in color space (for visualization we used the 2-D hue and saturation plane of the HSV color space). Crosses are the real generating values, black circles the current model values $\vec{T}_h$, and grey circles those of the previous iterations.

to $\sigma = 0.05$ and learning was initialized with $\hat{T}^{\text{init}} = \frac{1}{2}D$. The inferred approximate maximum-likelihood parameters converged to values close to the generating parameters in 44 of 50 trials. In 6 trials the algorithm represented 7 of the 8 causes. Its success rate, or *reliability*, is thus 88%. Fig. 3B shows the time-course of a typical trial during learning. As can be observed, the mask value $W$ and the feature values $T$ converged to values close to the generating ones. For data with added Gaussian pixel noise ($\sigma^{\text{gen}} = \sigma = 0.05$) the algorithms converges to values representing all causes in 48 of 50 trials (96% reliability). A higher average number of causes per input reduced reliability. A maximum of three causes (on average) were used for the noiseless bars test. This is considered a difficult task in the standard bars test. With otherwise the same parameters our algorithm had a reliability of 26% (50 trials) on this data. Performance seemed limited by the difficulty of the data rather than by the limitations of the used approximation. We could not increase the reliability of the algorithm when we increased the accuracy of (13) by setting $\chi = 4$ (instead of $\chi = 3$). Reliability seemed much more affected by changes to parameters for annealing and parameter noise, i.e., by changes to those parameters that affect the additional mechanisms to avoid local optima.

**The standard bars test.** Instead of choosing the bar colors randomly as above, they can also be set to specific values. In particular, if all bar colors are white, $\vec{T} = (1, 1, 1)^T$, the classical version of the bars test is recovered. Note that the learning algorithms can be applied to this standard form without modification. When the generating parameters were as above (eight bars, probability of a bar to be present $\frac{2}{8}$, $N = 500$), all bars were successfully extracted in 42 of 50 trials (84% reliability). For a bars test with ten bars, $D = 5 \times 5$, a probability of $\frac{2}{10}$ for each bar to be present, and $N = 500$ data points, the algorithm with model parameters as above extracted all bars in 43 of 50 trials (86% reliability; mean number of extracted bars 9.5). Reliability for this test increased when we increased the number of training images. For $N = 1000$ instead of 500 reliability increased to 94% (50 trials; mean number of extracted bars 9.9). The bars test with ten bars is probably the one most frequently found in the literature. Linear and non-linear component extraction approaches are compared, e.g., in [12, 8] and usually achieve lower reliability values than the presented algorithm. Classical ICA and PCA algorithms investigated in [13] never succeeded in extracting all bars. Relatively recent approaches can achieve reliability values higher than 90% but often only by introducing additional constraints (compare R-MCA [8], or constrained forms of NMF [14]).

**More realistic input.** One possible criticism of the bars tests above is that the bars are relatively simple objects. The purpose of this section is, therefore, to demonstrate the performance of the algorithm when images contain more complicated objects. Sized objects were taken from the COIL-100 dataset [15] with relatively uniform color distribution (objects 2, 4, 47, 78, 94, 97; all with zero degree rotation). The images were scaled down to $15 \times 15$ pixels and randomly placed on a black background image of $25 \times 25$ pixels. Downscaling introduced blurred object edges and to remove this effect dark pixels were set to black. The training images were generated with each object being

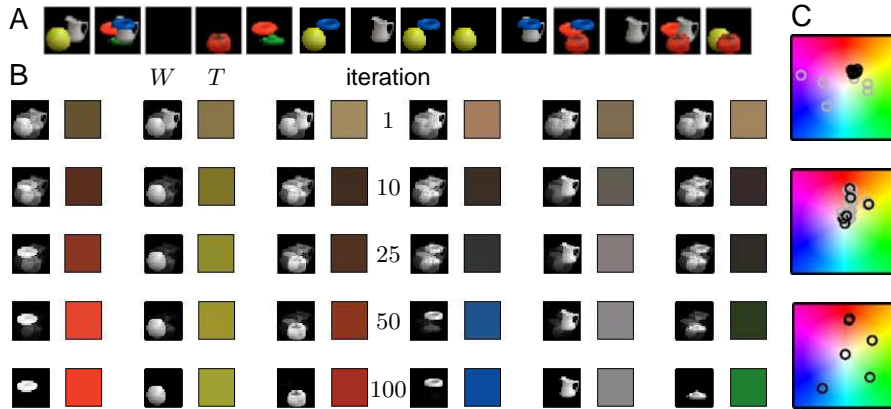

Figure 4: Application to images of cluttered objects. **A** Selection of $14$ of the $N = 500$ data points. **B** Parameter change displayed as in Fig. 3. **C** Change of feature vectors displayed as in Fig. 3.

present with probability $\frac{2}{6}$ and at a random depth. $N = 500$ such images were generated. Example images[1] are given in Fig. 4A. We applied the learning algorithm with $H = 6$, an initial temperature for annealing of $\hat{T}^{\text{init}} = \frac{1}{4}D$, and parameters as above otherwise. Fig. 4B shows the development of parameter values during learning. As can be observed, the mask values converged to represent the different objects, and the feature vectors converged to values representing the mean object color. Note that the model is not matched to the dataset as each object has a fixed distribution of color values which is a poor match to a Gaussian distribution with a constant color mean. The model reacted by assigning part of the real color distribution to the mask values which are responsible for the 3-dimensional appearance of the masks (see Fig. 4B). Note that the normalization (12) was motivated by this observation because it can better tolerate high mask value variances. We ran 50 trials using different sets of $N = 500$ images generated as above. In $42$ of the trials (84%) the algorithm converged to values representing all six objects together with appropriate values for their mean colors. In seven trials the algorithm converged to a local optima (average number of extracted objects was 5.8). In 50 trials with 8 objects (we added objects 36 and 77 of the COIL-100 database) an algorithm with same parameters but $H = 8$ extracted all objects in 40 of the trials (reliability 80%, average number of extracted objects 7.7).

## 5 Discussion

We have studied learning in the generative model of occlusion (1) to (4). Parameters can be optimized given a collection of $N$ images in which different sets of causes are present at different positions in depth. As briefly discussed earlier, the problem of occlusion has been addressed by other system before. E.g., the approach in [3, 4] uses a fixed number of layers, so called *sprites*, to model an order in depth. The approach assigns, to each pixel, probabilities that it has been generated by a specific sprite. Typically, the algorithms are applied to data which consist of images that have a small number of foreground objects (usually one or two) on a static or slowly changing background. Typical applications of the approach are figure-ground separation and the automatic removal of the background or foreground objects. The approach using sprites is in many aspects more general than the model presented in this paper. It includes, for instance, variable estimation for illumination and, importantly, addresses the problem of invariance by modeling object transformations. Regarding the modelling of object arrangements, our approach is, however, more general. The additional hidden variable used for object arrangements allows our model to be applied to images of cluttered scenes. The approach in [3, 4] assumes a fixed object arrangement, i.e., it assumes that each object has the same depth position in all training images. Our approach therefore addresses an aspect of visual data that is complementary to the aspects modeled in [3, 4]. Models that combine the advantages of

both approaches thus promise interesting advancements, e.g., towards systems that can learn from video data in which objects change their positions in depth.

Another interesting aspect of the model presented in this work is its close connection to component extraction methods. Algorithms such as SC, NMF or maximal causes analysis (MCA; [8]) use superpositions of elementary components to explain the data. ICA and SC have prominently been applied to explain neural response properties, and NMF is a popular approach to learn components for visual object recognition [e.g. 14, 16]. Our model follows these multiple-causes methods by assuming the data to consist of independently generated components. It distinguishes itself, however, by the way in which these components are assumed to combine. ICA, SC, NMF and many other models assume linear superposition, MCA uses a $\max$-function instead of the sum, and other systems use noisy-or combinations. In the class of multiple-causes approaches our model is the first to generalize the combination rule to one that models occlusion explicitly. This required an additional variable for depth and the introduction of two sets of parameters: masks and features. Note that in the context of multiple-causes models, masks have recently been introduced in conjunction with ICA [17] in order to model local contrast correlation in image patches. For our model, the combination of masks and vectorial feature parameters allow for applications to more general sets of data than those used for classical component extraction. In numerical experiments we have used color images for instance. However, we can apply our algorithm also to grey-level data such as used for other algorithms. This allows for a direct quantitative comparison of the novel algorithm with state-of-the-art component extraction approaches. The reported results for the standard bars test show the competitiveness of our approach despite its larger set of parameters [compare, e.g., 12, 8]. A limitation of the training method used is its assumption of relatively sparsely active hidden causes. This limitation is to some extent shared, e.g., with SC or sparse versions of NMF. Experiments with higher $\chi$ values in (13) indicate, however, that the performance of the algorithm is not so much limited by the accuracy of the E-step, but rather by the more challenging likelihood landscape for less sparse data.

For applications to visual data, color is the most straight-forward feature to model. Possible alternatives are, however, Gabor feature vectors which model object textures (see, e.g., [18] and references therein), SWIFT features [19], or vectors using combinations of color and texture [e.g. 6]. Depending on the choice of feature vectors and the application domain, it might be necessary to generalize the model. It is, for instance, straight-forward to introduce more complex feature vectors. Although one feature, e.g. one color, per cause can represent a suitable model for many applications, it can for other applications also make sense to use multiple feature vectors per cause. In the extreme case as many feature vectors as pixels could be used, i.e., $\vec{T}_h \rightarrow \vec{T}_{hd}$. The derivation of update rules for such features would proceed along the same lines as the derivations for single features $\vec{T}_h$. Furthermore, individual prior parameters for the frequency of object appearances could be introduced. Such parameters could be trained with an approach similar to the one in [8]. Additional parameters could also be introduced to model different prior probabilities for different arrangements in depth. An easy alteration would be, for instance, to always map one specific hidden unit to the most distant position in depth in order to model a background. Finally, the most interesting, but also most challenging generalization direction would be the inclusion of invariance principles. In its current form the model has, in common with state-of-the-art component extraction algorithms, the assumption that the component locations are fixed. Especially for images of objects, changes in planar component positions have to be addressed in general. Possible approaches that have been used in the literature can, for instance, be found in [3, 4] in the context of occlusion modeling, in [20] in the context of NMF, and in [18] in the context of object recognition. Potential future application domains for our approach would, however, also include data sets for which component positions are fixed. E.g., in many benchmark databases for face recognition, faces are already in a normalized position. For component extraction, faces can be regarded as combinations of a background faces 'occluded' by mouth, nose, and eye textures which can themselves be occluded by beards, sunglasses, or hats.

In summary, the studied occlusion model advances generative modeling approaches to visual data by explicitly modeling object arrangements in depth. The approach complements established approaches of occlusion modeling in the literature by generalizing standard approaches to multiple-causes component extraction.

**Acknowledgements.** We gratefully acknowledge funding by the German Federal Ministry of Education and Research (BMBF) in the project 01GQ0840 (Bernstein Focus Neurotechnology Frankfurt), the Gatsby Charitable Foundation, and the Honda Research Institute Europe GmbH.

## Footnotes

[1]Note that this appears much easier for a human observer because he/she can also make use of object knowlege, e.g., of the *gestalt* law of proximity. The difficulty of the data would become obvious if all pixels in each image of the data set were permuted by a fixed permutation map.

# References

[1] B. A. Olshausen and D. J. Field. Emergence of simple-cell receptive field properties by learning a sparse code for natural images. *Nature*, 381:607 – 609, 1996.

[2] D. D. Lee and H. S. Seung. Learning the parts of objects by non-negative matrix factorization. *Nature*, 401(6755):788–91, 1999.

[3] N. Jojic and B. Frey. Learning flexible sprites in video layers. *Conf. on Computer Vision and Pattern Recognition*, 1:199–206, 2001.

[4] C. K. I. Williams and M. K. Titsias. Greedy learning of multiple objects in images using robust statistics and factorial learning. *Neural Computation*, 16(5):1039–1062, 2004.

[5] K. Fukushima. Restoring partly occluded patterns: a neural network model. *Neural Networks*, 18(1):33–43, 2005.

[6] C. Eckes, J. Triesch, and C. von der Malsburg. Analysis of cluttered scenes using an elastic matching approach for stereo images. *Neural Computation*, 18(6):1441–1471, 2006.

[7] R. M. Neal and G. E. Hinton. A view of the EM algorithm that justifies incremental, sparse, and other variants. In M. I. Jordan, editor, *Learning in Graphical Models*. Kluwer, 1998.

[8] J. Lücke and M. Sahani. Maximal causes for non-linear component extraction. *Journal of Machine Learning Research*, 9:1227 – 1267, 2008.

[9] N. Ueda and R. Nakano. Deterministic annealing EM algorithm. *Neural Networks*, 11(2):271–282, 1998.

[10] M. Sahani. Latent variable models for neural data analysis, 1999. PhD Thesis, Caltech.

[11] P. Földiák. Forming sparse representations by local anti-Hebbian learning. *Biol Cybern*, 64:165 – 170, 1990.

[12] M. W. Spratling. Learning image components for object recognition. *Journal of Machine Learning Research*, 7:793 – 815, 2006.

[13] S. Hochreiter and J. Schmidhuber. Feature extraction through LOCOCODE. *Neural Computation*, 11:679 – 714, 1999.

[14] P. O. Hoyer. Non-negative matrix factorization with sparseness constraints. *Journal of Machine Learning Research*, 5:1457–1469, 2004.

[15] S. A. Nene, S. K. Nayar, and H. Murase. Columbia object image library (COIL-100). Technical report, cucs-006-96, 1996.

[16] H. Wersing and E. Körner. Learning optimized features for hierarchical models of invariant object recognition. *Neural Computation*, 15(7):1559–1588, 2003.

[17] U. Köster, J. T. Lindgren, M. Gutmann, and A. Hyvärinen. Learning natural image structure with a horizontal product model. In *Int. Conf. on Independent Component Analysis and Signal Separation (ICA)*, pages 507–514, 2009.

[18] P. Wolfrum, C. Wolff, J. Lücke, and C. von der Malsburg. A recurrent dynamic model for correspondence-based face recognition. *Journal of Vision*, 8(7):1–18, 2008.

[19] D. G. Lowe. Distinctive image features from scale-invariant keypoints. *International Journal of Computer Vision*, 60(2):91–110, 2004.

[20] J. Eggert, H. Wersing, and E. Körner. Transformation-invariant representation and NMF. In *Int. J. Conf. on Neural Networks (IJCNN)*, pages 2535–2539, 2004.

